# Faster Rates in Regression via Active Learning

**Rui Castro**
Rice University
Houston, TX 77005
rcastro@rice.edu

**Rebecca Willett**
University of Wisconsin
Madison, WI 53706
willett@cae.wisc.edu

**Robert Nowak**
University of Wisconsin
Madison, WI 53706
nowak@engr.wisc.edu

## Abstract

This paper presents a rigorous statistical analysis characterizing regimes in which active learning significantly outperforms classical passive learning. Active learning algorithms are able to make queries or select sample locations in an online fashion, depending on the results of the previous queries. In some regimes, this extra flexibility leads to significantly faster rates of error decay than those possible in classical passive learning settings. The nature of these regimes is explored by studying fundamental performance limits of active and passive learning in two illustrative nonparametric function classes. In addition to examining the theoretical potential of active learning, this paper describes a practical algorithm capable of exploiting the extra flexibility of the active setting and provably improving upon the classical passive techniques. Our active learning theory and methods show promise in a number of applications, including field estimation using wireless sensor networks and fault line detection.

## 1 Introduction

In this paper we address the theoretical capabilities of active learning for estimating functions in noise. Several empirical and theoretical studies have shown that selecting samples or making strategic queries in order to learn a target function/classifier can outperform commonly used passive methods based on random or deterministic sampling [1–5]. There are essentially two different scenarios in active learning: (i) *selective sampling*, where we are presented a pool of examples (possibly very large), and for each of these we can decide whether to collect a label associated with it, the goal being learning with the least amount of carefully selected labels [3]; (ii) *adaptive sampling*, where one chooses an experiment/sample location based on previous observations [4,6]. We consider adaptive sampling in this paper. Most previous analytical work in active learning regimes deals with very stringent conditions, like the ability to make perfect or nearly perfect decisions at every stage in the sampling procedure. Our working scenario is significantly less restrictive, and based on assumptions that are more reasonable for a broad range of practical applications.

We investigate the problem of nonparametric function regression, where the goal is to estimate a function from noisy point-wise samples. In the classical (passive) setting the sampling locations are chosen *a priori*, meaning that the selection of the sample locations precedes the gathering of the function observations. In the active sampling setting, however, the sample locations are chosen in an online fashion: the decision of where to sample

next depends on all the observations made previously, in the spirit of the "Twenty Questions" game (in passive sampling all the questions need to be asked before any answers are given). The extra degree of flexibility garnered through active learning can lead to significantly better function estimates than those possible using classical (passive) methods. However, there are very few analytical methodologies for these Twenty Questions problems when the answers are not entirely reliable (see for example [6–8]); this precludes performance guarantees and limits the applicability of many such methods. To address this critical issue, in this paper we answer several pertinent questions regarding the fundamental performance limits of active learning in the context of regression under noisy conditions.

Significantly faster rates of convergence are generally achievable in cases involving functions whose complexity (in a the Kolmogorov sense) is highly concentrated in small regions of space (*e.g.*, functions that are smoothly varying apart from highly localized abrupt changes such as jumps or edges). We illustrate this by characterizing the fundamental limits of active learning for two broad nonparametric function classes which map $[0, 1]^d$ onto the real line: (i) Hölder smooth functions (spatially homogeneous complexity) and (ii) piecewise constant functions that are constant except on a $d - 1$ dimensional *boundary set* or discontinuity embedded in the $d$ dimensional function domain (spatially concentrated complexity). The main result of this paper is two-fold. First, when the complexity of the function is spatially homogeneous, passive learning algorithms are near-minimax optimal over all estimation methods and all (active or passive) learning schemes, indicating that active learning methods cannot provide faster rates of convergence in this regime. Second, for piecewise constant functions, active learning methods can capitalize on the highly localized nature of the boundary by focusing the sampling process in the estimated vicinity of the boundary. We present an algorithm that provably improves on the best possible passive learning algorithm and achieves faster rates of error convergence. Furthermore, we show that this performance cannot be significantly improved on by any other active learning method (in a minimax sense). Earlier existing work had focused on one dimensional problems [6, 7], and very specialized multidimensional problems that can be reduced to a series of one dimensional problems [8]. Unfortunately these techniques cannot be extended to more general piecewise constant/smooth models, and to the best of our knowledge our work is the first addressing active learning in this class of models.

Our active learning theory and methods show promise for a number of problems. In particular, in imaging techniques such as laser scanning it is possible to adaptively vary the scanning process. Using active learning in this context can significantly reduce image acquisition times. Wireless sensor network constitute another key application area. Because of necessarily small batteries, it is desirable to limit the number of measurements collected as much as possible. Incorporating active learning strategies into such systems can dramatically lengthen the lifetime of the system. In fact, active learning problems like the one we pose in Section 4 have already found application in fault line detection [7] and boundary estimation in wireless sensor networking [9].

## 2  Problem Statement

Our goal is to estimate $f : [0, 1]^d \to \mathbb{R}$ from a finite number of noise-corrupted samples. We consider two different scenarios: (a) *passive learning*, where the location of the sample points is chosen statistically independently of the measurement outcomes; and (b) *active learning*, where the location of the $i^{th}$ sample point can be chosen as a function of the samples points and samples collected up to that instant. The statistical model we consider builds on the following assumptions:

(A1) The observations $\{Y_i\}_{i=1}^n$ are given by
$$Y_i = f(\boldsymbol{X_i}) + W_i, \ i \in \{1, \ldots, n\}.$$

**(A2)** The random variables $W_i$ are Gaussian zero mean and variance $\sigma^2$. These are independent and identically distributed (i.i.d.) and independent of $\{\boldsymbol{X_i}\}_{i=1}^n$.

**(A3.1) Passive Learning:** The sample locations $\boldsymbol{X_i} \in [0,1]^d$ are either deterministic or random, but independent of $\{Y_j\}_{j \neq i}$. They do not depend in any way on $f$.

**(A3.2) Active Learning:** The sample locations $\boldsymbol{X_i}$ are random, and depend only on $\{\boldsymbol{X_j}, Y_j\}_{j=1}^{i-1}$. In other words the sample locations $\boldsymbol{X_i}$ have only a causal dependency on the system variables $\{\boldsymbol{X_i}, Y_i\}$. Finally, given $\{\boldsymbol{X_j}, Y_j\}_{j=1}^{i-1}$ the random variable $\boldsymbol{X_i}$ does not depend in any way on $f$.

Let $\hat{f}_n : [0,1]^d \to \mathbb{R}$ denote an estimator based on the training samples $\{\boldsymbol{X}_i, Y_i\}_{i=1}^n$. When constructing an estimator under the active learning paradigm there is another degree of freedom: we are allowed to choose our *sampling strategy*, that is, we can specify $\boldsymbol{X_i}|\boldsymbol{X_1} \ldots \boldsymbol{X_{i-1}}, Y_1 \ldots Y_{i-1}$. We will denote the sampling strategy by $S_n$. The pair $(\hat{f}_n, S_n)$ is called the *estimation strategy*. Our goal is to construct estimation strategies which minimize the expected squared error,

$$\mathbb{E}_{f,S_n}[\|\hat{f}_n - f\|^2],$$

where $\mathbb{E}_{f,S_n}$ is the expectation with respect to the probability measure of $\{\boldsymbol{X_i}, Y_i\}_{i=1}^n$ induced by model $f$ and sampling strategy $S_n$, and $\|\cdot\|$ is the usual $L_2$ norm.

## 3 Learning in Classical Smoothness Spaces

In this section we consider classes of functions whose complexity is homogeneous over the entire domain, so that there are no localized features, as in Figure 1(a). In this case we do not expect the extra flexibility of the active learning strategies to provide any substantial benefit over passive sampling strategies, since a simple uniform sampling scheme is naturally matched to the homogeneous "distribution" of the target function's complexity. To exemplify this consider the Hölder smooth function class: a function $f : [0,1]^d \to \mathbb{R}$ is *Hölder smooth* if it has continuous partial derivatives up to order $k = \lfloor \alpha \rfloor$ [1] and

$$\forall \, \boldsymbol{z}, \boldsymbol{x} \in [0,1]^d : \quad |f(\boldsymbol{z}) - P_{\boldsymbol{x}}(\boldsymbol{z})| \leq L\|\boldsymbol{z} - \boldsymbol{x}\|^\alpha,$$

where $L, \alpha > 0$, and $P_{\boldsymbol{x}}(\cdot)$ denotes the order $k$ Taylor polynomial approximation of $f$ expanded around $\boldsymbol{x}$. Denote this class of functions by $\Sigma(L, \alpha)$. Functions in $\Sigma(L, \alpha)$ are essentially $C^\alpha$ functions when $\alpha \in \mathbb{N}$. The first of our two main results is a minimax lower bound on the performance of all active estimation strategies for this class of functions.

**Theorem 1.** *Under the requirements of the active learning model we have the minimax bound*

$$\inf_{(\hat{f}_n, S_n) \in \Theta_{active}} \sup_{f \in \Sigma(L,\alpha)} \mathbb{E}_{f,S_n}[\|\hat{f}_n - f\|^2] \geq cn^{-\frac{2\alpha}{2\alpha+d}}, \tag{1}$$

*where $c \equiv c(L, \alpha, \sigma^2) > 0$ and $\Theta_{active}$ is the set of all active estimation strategies (which includes also passive strategies).*

Note that the rate in Theorem 1 is the same as the classical passive learning rate [10, 11] but the class of estimation strategies allowed is now much bigger. The proof of Theorem 1 is presented in our technical report [12] and uses standard tools of minimax analysis, such as Assouad's Lemma. The key idea of the proof is to reduce the problem of estimating a function in $\Sigma(L, \alpha)$ to the problem of deciding among a finite number of hypotheses. The key aspects of the proof for the passive setting [13] apply to the active scenario due to the fact that we can choose an adequate set of hypotheses without knowledge of the sampling strategy, although some modifications are required due to the extra flexibility of the sampling strategy. There are various practical estimators achieving the performance predicted by Theorem 1, including some based on kernels, splines or wavelets [13].

## 4  The Active Advantage

In this section we address two key questions: (i) when does active learning provably yield better results, and (ii) what are the fundamental limitations of active learning? These are difficult questions to answer in general. We expect that, for functions whose complexity is spatially non-uniform and highly concentrated in small subsets of the domain, the extra spatial adaptivity of the active learning paradigm can lead into significant performance gains. We study a class of functions which highlights this notion of "spatially concentrated complexity". Although this is a canonical example and a relatively simple function class, it is general enough to provide insights into methodologies for broader classes.

A function $f : [0,1]^d \to \mathbb{R}$ is called *piecewise constant* if it is locally constant[2] in any point $\boldsymbol{x} \in [0,1]^d \setminus B(f)$, where $B(f) \subset [0,1]^d$, the *boundary set*, has upper box-counting dimension at most $d - 1$. Furthermore let $f$ be uniformly bounded on $[0,1]^d$ (that is, $|f(\boldsymbol{x})| \leq M, \ \forall \boldsymbol{x} \in [0,1]^d$) and let $B(f)$ satisfy $N(r) \leq \beta r^{-(d-1)}$ for all $r > 0$, where $\beta > 0$ is a constant and $N(r)$ is the minimal number of closed balls of diameter $r$ that covers $B(f)$. The set of all piecewise constant functions $f$ satisfying the above conditions is denoted by $\mathrm{PC}(\beta, M)$.

The conditions above mean that (a) the functions are constant except along $d - 1$-dimensional "boundaries" where they are discontinuous and (b) the boundaries between the various constant regions are $(d - 1)$-dimensional non-fractal sets. If the boundaries $B(f)$ are smooth then $\beta$ is an approximate bound on their total $d - 1$ dimensional volume (*e.g.*, the length if $d = 2$). An example of such a function is depicted in Figure 1(b). The class $\mathrm{PC}(\beta, M)$ has the main ingredients that make active learning appealing: a function $f$ is "well-behaved" everywhere on the unit square, except on a small subset $B(f)$. We will see that the critical task for any good estimator is to accurately find the location of the boundary $B(f)$.

### 4.1  Passive Learning Framework

To obtain minimax lower bounds for $\mathrm{PC}(\beta, M)$ we consider a smaller class of functions, namely the boundary fragment class studied in [11]. Let $g : [0,1]^{d-1} \to [0,1]$ be a Lipshitz function with graph in $[0,1]^d$, that is

$$|g(\boldsymbol{x}) - g(\boldsymbol{z})| \leq \|\boldsymbol{x} - \boldsymbol{z}\|, \ 0 \leq g(\boldsymbol{x}) \leq 1, \ \forall \, \boldsymbol{x}, \boldsymbol{z} \in [0,1]^{d-1}.$$

Define $G = \{(\boldsymbol{x}, y) : 0 \leq y \leq g(\boldsymbol{x}), \ \boldsymbol{x} \in [0,1]^{d-1}\}$. Finally define $f : [0,1]^d \to \mathbb{R}$ by $f(\boldsymbol{x}) = 2M\mathbf{1}_G(\boldsymbol{x}) - M$. The class of all the functions of this form is called the *boundary fragment* class (usually $M = 1$), denoted by $\mathrm{BF}(M)$. Note that there are only two regions, and the boundary separating those is a function of the first $d - 1$ variables.

It is straightforward to show that $\mathrm{BF}(M) \subseteq \mathrm{PC}(\beta, M)$ for a suitable constant $\beta$; therefore a minimax lower bound for the boundary fragment class is trivially a lower bound for the piecewise constant class. From the results in [11] we have

$$\inf_{(\hat{f}_n, S_n) \in \Theta_{\text{passive}}} \sup_{f \in \mathrm{PC}(\beta, M)} \mathbb{E}_{f, S_n}[d^2(\hat{f}_n, f)] \geq cn^{-\frac{1}{d}}, \tag{2}$$

where $c \equiv c(\beta, M, \sigma^2) > 0$.

There exist practical passive learning strategies that are near-minimax optimal. For example, tree-structured estimators based on *Recursive Dyadic Partitions* (RDPs) are capable of

$$\exists \epsilon > 0 \ : \forall \boldsymbol{y} \in [0,1]^d : \quad \|\boldsymbol{x} - \boldsymbol{y}\| < \epsilon \ \Rightarrow \ f(\boldsymbol{y}) = f(\boldsymbol{x}).$$

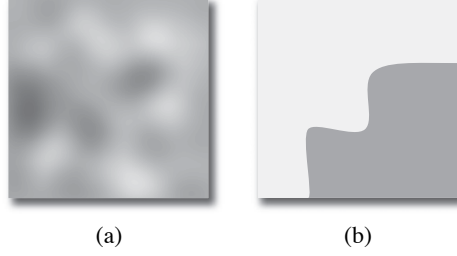

(a)                    (b)

Figure 1: Examples of functions in the classes considered: (a) Hölder smooth function. (b) Piecewise constant function.

nearly attaining the minimax rate above [14]. These estimators are constructed as follows: (i) Divide $[0,1]^d$ into $2^d$ equal sized hypercubes. (ii) Repeat this process again on each hypercube. Repeating this process $\log_2 m$ times gives rise to a partition of the unit hypercube into $m^d$ hypercubes of identical size. This process can be represented as a $2^d$-ary tree structure (where a leaf of the tree corresponds to a partition cell). Pruning this tree gives rise to an RDP with non-uniform resolution. Let $\Pi$ denote the class of all possible pruned RDPs. The estimators we consider are constructed by decorating the elements of a partition with constants. Let $\pi$ be an RDP; the estimators built over this RDP have the form $\tilde{f}^{(\pi)}(\boldsymbol{x}) \equiv \sum_{A\in\pi} c_A \mathbf{1}\{x \in A\}$.

Since the location of the boundary is *a priori* unknown it is natural to distribute the sample points uniformly over the unit cube. There are various ways of doing this; for example, the points can be placed deterministically over a lattice, or randomly sampled from a uniform distribution. We will use the latter strategy. Assume that $\{\boldsymbol{X}_i\}_{i=1}^n$ are i.i.d. uniform over $[0,1]^d$. Define the *complexity regularized estimator* as

$$\hat{f}_n \equiv \arg \min_{\tilde{f}^{(\pi)}:\pi\in\Pi} \left\{ \frac{1}{n}\sum_{i=1}^n \left(\tilde{f}^{(\pi)}(\boldsymbol{X}_i)-Y_i\right)^2 + \lambda \frac{\log n}{n}|\pi| \right\}, \qquad (3)$$

where $|\pi|$ denotes the number of elements of $\pi$ and $\lambda > 0$. The above optimization can be solved efficiently in $O(n)$ operations using a bottom-up tree pruning algorithm [14].

The performance of the estimator in (3) can be assessed using bounding techniques in the spirit of [14,15]. From that analysis we conclude that

$$\sup_{f\in\mathrm{PC}(\beta,M)} \mathbb{E}_f[\|\hat{f}_n - f\|^2] \leq C(n/\log n)^{-\frac{1}{d}}, \qquad (4)$$

where $C \equiv C(\beta, M, \sigma^2) > 0$. This shows that, up to a logarithmic factor, the rate in (2) is the optimal rate of convergence for passive strategies. A complete derivation of the above result is available in [12].

## 4.2   Active Learning Framework

We now turn our attention to the active learning scenario. In [8] this was studied for the boundary fragment class. From that work and noting again that $\mathrm{BF}(M) \subseteq \mathrm{PC}(\beta, M)$ we have, for $d \geq 2$,

$$\inf_{(\hat{f}_n,S_n)\in\Theta_{\mathrm{active}}} \sup_{f\in\mathrm{PC}(\beta,M)} \mathbb{E}_{f,S_n}[\|\hat{f}_n - f\|^2] \geq cn^{-\frac{1}{d-1}}, \qquad (5)$$

where $c \equiv c(M, \sigma^2) > 0$.

In contrast with (2), we observe that with active learning we have a potential performance gain over passive strategies, effectively equivalent to a dimensionality reduction. Essentially the exponent in (5) depends now on the dimension of the boundary set, $d-1$, instead

of the dimension of the entire domain, $d$. In [11] an algorithm capable of achieving the above rate for the boundary fragment class is presented, but this algorithm takes advantage of the very special functional form of the boundary fragment functions. The algorithm begins by dividing the unit hypercube into "strips" and performing a one-dimensional change-point estimation in each of the strips. This change-point detection can be performed extremely accurately using active learning, as shown in the pioneering work of Burnashev and Zigangirov [6]. Unfortunately, the boundary fragment class is very restrictive and impractical for most applications. Recall that boundary fragments consist of only two regions, separated by a boundary that is a function of the first $d-1$ coordinates. The class $\mathrm{PC}(\beta, M)$ is much larger and more general and the algorithmic ideas that work for boundary fragments can no longer be used. A completely different approach is required, using radically different tools.

We now propose an active learning scheme for the piecewise constant class. The proposed scheme is a two-step approach based in part on the tree-structured estimators described above for passive learning. In the first step, called the *preview step*, a rough estimator of $f$ is constructed using $n/2$ samples (assume for simplicity that $n$ is even), distributed uniformly over $[0, 1]^d$. In the second step, called the *refinement step*, we select $n/2$ samples near the perceived locations of the boundaries (estimated in the preview step) separating constant regions. At the end of this process we will have half the samples concentrated in the vicinity of the boundary set $B(f)$. Since accurately estimating $f$ near $B(f)$ is key to obtaining faster rates, the strategy described seems quite sensible. However, it is *critical* that the preview step is able to detect the boundary with very high probability. If part of the boundary is missed, then the error incurred is going to propagate into the final estimate, ultimately degrading the performance. Therefore extreme care must be taken to detect the boundary in the preview step, as described below.

**Preview:** The goal of this stage is to provide a coarse estimate of the location of $B(f)$. Specifically, collect $n' \equiv n/2$ samples at points distributed uniformly over $[0, 1]^d$. Next proceed by using the passive learning algorithm described before, but restrict the estimator to RDPs with leafs at a maximum depth of $J = \frac{d-1}{(d-1)^2+d} \log(n'/\log(n'))$. This ensures that, on average, every element of the RDP contains many sample points; therefore we obtain a low variance estimate, although the estimator bias is going to be large. In other words, we obtain a very "stable" coarse estimate of $f$, where stable means that the estimator does not change much for different realizations of the data.

The above strategy ensures that most of the time, leafs that intersect the boundary are at the maximum allowed depth (because otherwise the estimator would incur too much empirical error) and leafs away from the boundary are at shallower depths. Therefore we can "detect" the rough location of the boundary just by looking at the deepest leafs. Unfortunately, if the set $B(f)$ is somewhat aligned with the dyadic splits of the RDP, leafs intersecting the boundary can be pruned without incurring a large error. This is illustrated in Figure 2(b); the cell with the arrow was pruned and contains a piece of the boundary, but the error incurred by pruning is small since that region is mostly a constant region. However, worst-case analysis reveals that the squared bias induced by these small volumes can add up, precluding the desired rates. A way of mitigating this issue is to consider multiple RDP-based estimators, each one using RDPs appropriately shifted. We use $d+1$ estimators in the preview step: one on the initial uniform partition, and $d$ over partitions whose dyadic splits have been translated by $2^{-J}$ in each one of the $d$ coordinates. Any leaf that is at the maximum depth of any of the $d+1$ RDPs pruned in the preview step indicates the highly probable presence of a boundary, and will be refined in the next stage.

**Refinement:** With high probability, the boundary is contained in the leafs at the maximum depth. In the refinement step we collect additional $n/2$ samples in the corresponding partition cells, using these to obtain a refined estimate of the function $f$ by again applying

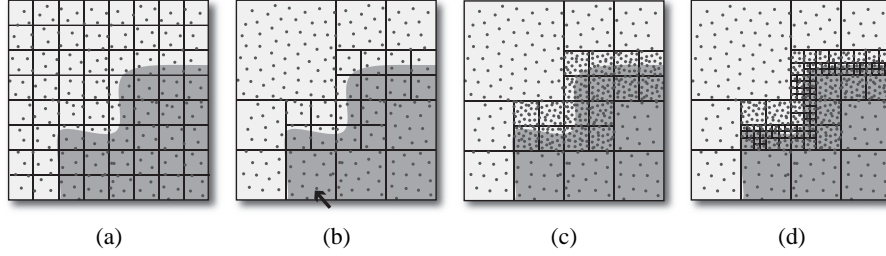

(a)          (b)          (c)          (d)

Figure 2: The two step procedure for $d = 2$: (a) Initial unpruned RDP and $n/2$ samples. (b) Preview step RDP. Note that the cell with the arrow was pruned, but it contains a part of the boundary. (c) Additional sampling for the refinement step. (d) Refinement step.

an RDP-based estimator. This produces a higher resolution estimate in the vicinity of the boundary set $B(f)$, yielding better performance than the passive learning technique.

To formally show that this algorithm attains the faster rates we desire we have to consider a further technical assumption, namely that the boundary set is "cusp-free"[3]. This condition is rather technical, but it is not very restrictive, and encompasses many interesting situations, including of course boundary fragments. For a more detailed explanation see [12]. Under this condition we have the following:

**Theorem 2.** *Under the active learning scenario we have, for $d \geq 2$ and functions $f$ whose boundary is cusp-free,*

$$\mathbb{E}\left[\|\hat{f}_n - f\|^2\right] \quad \leq C \left(\frac{n}{\log n}\right)^{-\frac{1}{d-1+1/d}}, \tag{6}$$

*where $C > 0$.*

This bound improves on (4), demonstrating that this technique performs better than the best possible passive learning estimator. The proof of Theorem 2 is quite involved and is presented in detail in [12]. The main idea behind the proof is to decompose the error of the estimator for three different cases: (i) the error incurred during the preview stage in regions "away" from the boundary; (ii) the error incurred by not detecting a piece of the boundary (and therefore not performing the refinement step in that area); (iii) the error remaining in the refinement region at the end of the process. By restricting the maximum depth of the trees in the preview stage we can control the type-(i) error, ensuring that it does not exceed the error rate in (6). Type-(ii) error corresponds to the situations when a part of the boundary was not detected in the preview step. This can happen because of the inherent randomness of the noise and sampling distribution, or because the boundary is somewhat aligned with the dyadic splits. The latter can be a problem and this is why one needs to perform $d + 1$ preview estimates over shifted partitions. If the boundary is cusp-free then it is guaranteed that one of those preview estimators is going to "feel" the boundary since it is not aligned with the corresponding partition. Finally, the type-(iii) error is very easy to analyze, using the same techniques we used for the passive estimator.

A couple of remarks are important at this point. Instead of a two-step procedure one can reiterate this idea, performing multiple steps (*e.g.*, for a three-step approach replace the refinement step with the two-step approach described above). Doing so can further improve the performance. One can show that the expected error will decay like $n^{-1/(d-1+\epsilon)}$, with $\epsilon > 0$, given a sufficiently large number of steps. Therefore we can get rates arbitrarily close to the lower bound rates in (5).

## 5 Final Remarks

The results presented in this paper show that in certain scenarios active learning attains provable gains over the classical passive approaches. Active learning is an intuitively appealing idea and may find application in many practical problems. Despite these draws, the analysis of such active methods is quite challenging due to the loss of statistical independence in the observations (recall that now the sample locations are coupled with all the observations made in the past). The two function classes presented are non-trivial canonical examples illustrating under what conditions one might expect active learning to improve rates of convergence. The algorithm presented here for actively learning members of the piecewise constant class demonstrates the possibilities of active learning. In fact, this algorithm has already been applied in the context of field estimation using wireless sensor networks [9]. Future work includes the further development of the ideas presented here to the context of binary classification and active learning of the Bayes decision boundary.

## Footnotes

[1] $k = \lfloor \alpha \rfloor$ is the maximal integer such that $k < \alpha$.

[2]A function $f : [0,1]^d \to \mathbb{R}$ is locally constant at a point $\boldsymbol{x} \in [0,1]^d$ if

[3]A cusp-free boundary cannot have the behavior you observe in the graph of $|x|^{1/2}$ at the origin. Less "aggressive" kinks are allowed, such as in the graph of $|x|$.

## References

[1] D. Cohn, Z. Ghahramani, and M. Jordan, "Active learning with statistical models," *Journal of Artificial Intelligence Research*, pp. 129–145, 1996.

[2] D. J. C. Mackay, "Information-based objective functions for active data selection," *Neural Computation*, vol. 4, pp. 698–714, 1991.

[3] Y. Freund, H. S. Seung, E. Shamir, and N. Tishby, "Information, prediction, and query by committee," *Proc. Advances in Neural Information Processing Systems*, 1993.

[4] K. Sung and P. Niyogi, "Active learning for function approximation," *Proc. Advances in Neural Information Processing Systems*, vol. 7, 1995.

[5] G. Blanchard and D. Geman, "Hierarchical testing designs for pattern recognition," to appear in Annals of Statistics, 2005.

[6] M. V. Burnashev and K. Sh. Zigangirov, "An interval estimation problem for controlled observations," *Problems in Information Transmission*, vol. 10, pp. 223–231, 1974.

[7] P. Hall and I. Molchanov, "Sequential methods for design-adaptive estimation of discontinuities in regression curves and surfaces," *The Annals of Statistics*, vol. 31, no. 3, pp. 921–941, 2003.

[8] Alexander Korostelev, "On minimax rates of convergence in image models under sequential design," *Statistics & Probability Letters*, vol. 43, pp. 369–375, 1999.

[9] R. Willett, A. Martin, and R. Nowak, "Backcasting: Adaptive sampling for sensor networks," in *Proc. Information Processing in Sensor Networks*, 26-27 April, Berkeley, CA, USA, 2004.

[10] Charles J. Stone, "Optimal rates of convergence for nonparametric estimators," *The Annals of Statistics*, vol. 8, no. 6, pp. 1348–1360, 1980.

[11] A.P. Korostelev and A.B. Tsybakov, *Minimax Theory of Image Reconstruction*, Springer Lecture Notes in Statistics, 1993.

[12] R. Castro, R. Willett, and R. Nowak, "Fast rates in regression via active learning," Tech. Rep., University of Wisconsin, Madison, June 2005, ECE-05-3 Technical Report (available at http://homepages.cae.wisc.edu/ rcastro/ECE-05-3.pdf).

[13] Alexandre B. Tsybakov, *Introduction à l'estimation non-paramétrique*, Mathématiques et Applications, 41. Springer, 2004.

[14] R. Nowak, U. Mitra, and R. Willett, "Estimating inhomogeneous fields using wireless sensor networks," *IEEE Journal on Selected Areas in Communication*, vol. 22, no. 6, pp. 999–1006, 2004.

[15] Andrew R. Barron, "Complexity regularization with application to artificial neural networks," in *Nonparametric Functional Estimation and Related Topics*. 1991, pp. 561–576, Kluwer Academic Publishers.
